# Destabilization and Route to Chaos in Neural Networks with Random Connectivity

**Bernard Doyon**
Unité INSERM 230
Service de Neurologie
CHU Purpan
F-31059 Toulouse Cedex, France

**Bruno Cessac**
Centre d'Etudes et de Recherches
de Toulouse
2, avenue Edouard Belin, BP 4025
F-31055 Toulouse Cedex, France

**Mathias Quoy**
Centre d'Etudes et de Recherches
de Toulouse
2, avenue Edouard Belin, BP 4025
F-31055 Toulouse Cedex, France

**Manuel Samuelides**
Ecole Nationale Supérieure
de l'Aéronautique et de l'Espace
10, avenue Edouard Belin, BP 4032
F-31055 Toulouse Cedex, France

## Abstract

The occurence of chaos in recurrent neural networks is supposed to depend on the architecture and on the synaptic coupling strength. It is studied here for a randomly diluted architecture. By normalizing the variance of synaptic weights, we produce a bifurcation parameter, dependent on this variance and on the slope of the transfer function but independent of the connectivity, that allows a sustained activity and the occurence of chaos when reaching a critical value. Even for weak connectivity and small size, we find numerical results in accordance with the theoretical ones previously established for fully connected infinite sized networks. Moreover the route towards chaos is numerically checked to be a quasi-periodic one, whatever the type of the first bifurcation is (Hopf bifurcation, pitchfork or flip).

# 1    INTRODUCTION

Most part of studies on recurrent neural networks assume sufficient conditions of convergence. Models with symmetric synaptic connections have dynamical properties strongly connected with those of spin-glasses. In particular, they have relaxationnal dynamics caracterised by the decreasing of a function which is analogous to the energy in spin-glasses (or free energy for models submitted to thermal noise). Networks with asymmetric synaptic connections lose this convergence property and can have more complex dynamics, but searchers try to obtain such a convergence because the relaxation to a stable network state is simply interpreted as a stored pattern.

However, as pointed out by Hirsch (1989), it might be very interesting, from an engineering point of view, to investigate non convergent networks because their dynamical possibilities are much richer for a given number of units. Moreover, the real brain is a highly dynamic system. Recent neurophysiological findings have focused attention on the rich temporal structures (oscillations) of neuronal processes (Gray *et al.*, 1989), which might play an important role in information processing. Chaotic behavior has been found out in the nervous system (Gallez & Babloyantz, 1991) and might be implicated in cognitive processes (Skarda & Freeman, 1987).

We have studied the emergent dynamics of a general class of non convergent networks. Some results are already available in this field. Sompolinsky *et al.* (1988) established strong theoretical results concerning the occurrence of chaos for *fully connected networks* in the thermodynamic limit ($N \rightarrow \infty$) by using the Dynamic Mean Field Theory. Their model is a continuous time, continuous state dynamical system with $N$ fully connected neurons. Each connection $J_{ij}$ is a gaussian random variable with zero mean and a *normalized* variance $J^2/N$. As the $J_{ij}$'s are independent, the constant term $J^2$ can be seen as the variance of the sum of the weights connected to a given unit. Thus, the global strength of coupling remains constant for each neuron as $N$ increases. The output function of each neuron is sigmoidal with a slope $g$. Sompolinsky *et al.* established that, in the limit $N \rightarrow \infty$, there is a sharp transition from a stationary state to a chaotic flow. The onset of chaos is given by the critical value $gJ=1$. For $gJ<1$ the system admits the only fixed point zero, while for $gJ >1$ it is chaotic. The same authors performed simulations on finite and large values of $N$ and showed the existence of an intermediate regime (nonzero stationary states or limit cycles) separating the stationary and the chaotic phase, but the routes to chaos were not systematically explored. The range of $gJ$ where this intermediate behavior is observed shrinks as $N$ increases.

# 2    THE MODEL

The hypothesis of a fully connected network being not biologically plausible, it could be interesting to inspect how far these results could be extended as the dilution increases for a general class of networks. The model we study is defined as follows : the number of units is $N$, and $K$ is the fixed number of connections received by one unit ($K>1$). There is no connection from one unit to itself. The $K$ connections are randomly selected (with an uniform law) among the $N$-1's. The state of each neuron $i$ at time $t$ is characterized by its

output $x_i(t)$ which is a real variable varying between -1 and 1. The discrete and parallel dynamics is given by :

$$x_i(t+1) = tanh\left(g\sum_j J_{ij}\, x_j(t)\right)$$

$J_{ij}$ is the synaptic weight which couples the output of unit $j$ to the input of unit $i$. These weights are random independent variables chosen with a uniform law, with zero mean and a *normalized* variance $J^2/K$. Notice that, with such a normalization, the standard deviation of the sum of the weights afferent to a given neuron is the constant $J$.

One has to distinguish two effects of coupling on the behavior of such a class of models. The first effect is due to the strength of coupling, independent of the number of connections. The second one is due to the architecture of coupling, which can be studied by keeping constant the global synaptic effect of coupling. The genericity of our model cancels the peculiar dynamic features which may occur due to geometrical effects. Moreover it allows to study a model at different scales of dilution.

## 3   FIRST BIFURCATION

For such a system, zero is always a fixed point and for low bifurcation parameter value it is the only fixed point and it is stable. Let us call $\lambda_{max}$ the eigenvalue of the matrix of synaptic weights with the greatest modulus and $\rho = |\lambda_{max}|$ the spectral radius of this matrix. The loss of stability arises when the product $g\rho$ is larger than 1. Our numerical simulations allow us to state that $\rho$ *is approximately equal to J* for sufficiently large-sized networks. This statement can be derived rigorously for an approximate regularized model in the thermodynamic limit (Doyon *et al.*, 1993).

Table 1: Mean Value of the Bifurcation Parameter $gJ$ over 30 Networks.
Destabilization of the zero fixed point / Onset of Chaos

| Connectivity $K$ | Number of neurons | | |
|---|---|---|---|
| | 128 | 256 | 512 |
| 4 | .954 / 1.337 | .965 / 1.298 | .970 / 1.258 |
| 8 | .950 / 1.449 | .966 / 1.301 | .978 / 1.233 |
| 16 | .951 / 1.434 | .965 / 1.315 | .969 / 1.239 |
| 32 | .961 / 1.360 | .958 / 1.333 | .972 / 1.246 |

We have studied by intensive simulations on a Cray I-XMP computer the statistical spectral distribution for $N$ ranging from 4 to 512 and for $K$ ranging from 2 to 32. Figure 1 shows two examples of spectra (for convenience, $J$ is set to 1). The apparent drawing of a real axis is due to the real eigenvalue density but the distribution converges to a uniform one over the $J$ radius disk, as $N$ increases. A similar result has been theoretically

achieved for full gaussian matrices (Girko, 1985 ; Sommers *et al.*, 1988). Thus $\rho$ quickly decreases to $J$, so the loss of stability arises for a mean $gJ$ value that increases to 1 for increasing size (Tab. 1). For a given $N$ value, $\rho$ is nearly independent of $K$.

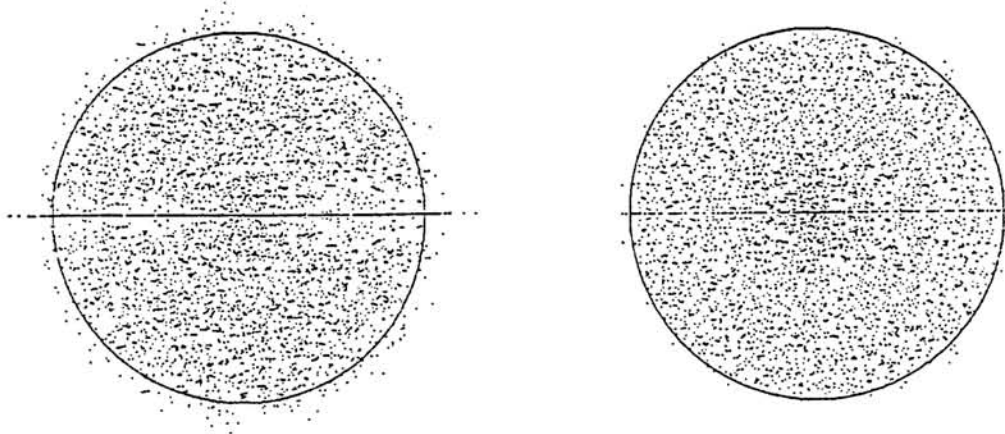

Figure 1: Plot of the Unit Disk and of the Eigenvalues in the Complex Plane.
Left: 100 Spectra for $N$=64, $K$=4. Right: 10 Spectra for $N$=512, $K$=4.

Three types of first bifurcation can occur, depending on the eigenvalue $\lambda_{max}$ :

a) *Hopf Bifurcation*: this corresponds to the appearance of oscillations. There are two complex conjugate eigenvalues with maximal modulus $\rho$.

b) *Pitchfork bifurcation*: if $\lambda_{max}$ is real positive, the bifurcation arises when $g\lambda_{max} = 1$. Zero loses its stability and two branches of stable equilibria emerge.

c) *Flip Bifurcation*: for $\lambda_{max}$ real and negative a flip bifurcation occurs when $g\lambda_{max} = -1$. This corresponds to the appearance of a period two oscillation.

As the network size increases, the proportion of Hopf bifurcations increases because the proportion of real $\lambda_{max}$ decreases, nearly independent of $K$.

## 4   ROUTE TO CHAOS

To study the following bifurcations, we chose the global observable:

$$m(t) = \frac{1}{N} \sum_{i=1}^{N} x_i (t)$$

The value $m(t)$ correctly characterizes all types of first bifurcation that can occur. Indeed the route to chaos is *qualitatively* well described by this observable, as we checked it by

studying simultaneously $x_i(t)$. The onset of chaos was computed by testing the sensitivity on initial conditions for $m(t)$. We observed the onset of chaos occurs for quite low parameter values. The transient zone from fixed point to chaos shrinks slowly to zero as the network size increases (Tab. 1).

The qualitative study of the routes to chaos was made on a span of networks with various connectivity and quite important size. The route towards chaos that was observed was a quasi-periodic one in all cases with some variations due to the particular symmetry $x \rightarrow -x$. The following figures are obtained by plotting $m(t+1)$ versus $m(t)$ after discarding the transient (Fig. 2). They are not qualitatively different with a reconstruction in a higher dimensional space. The dominant features are the following ones.

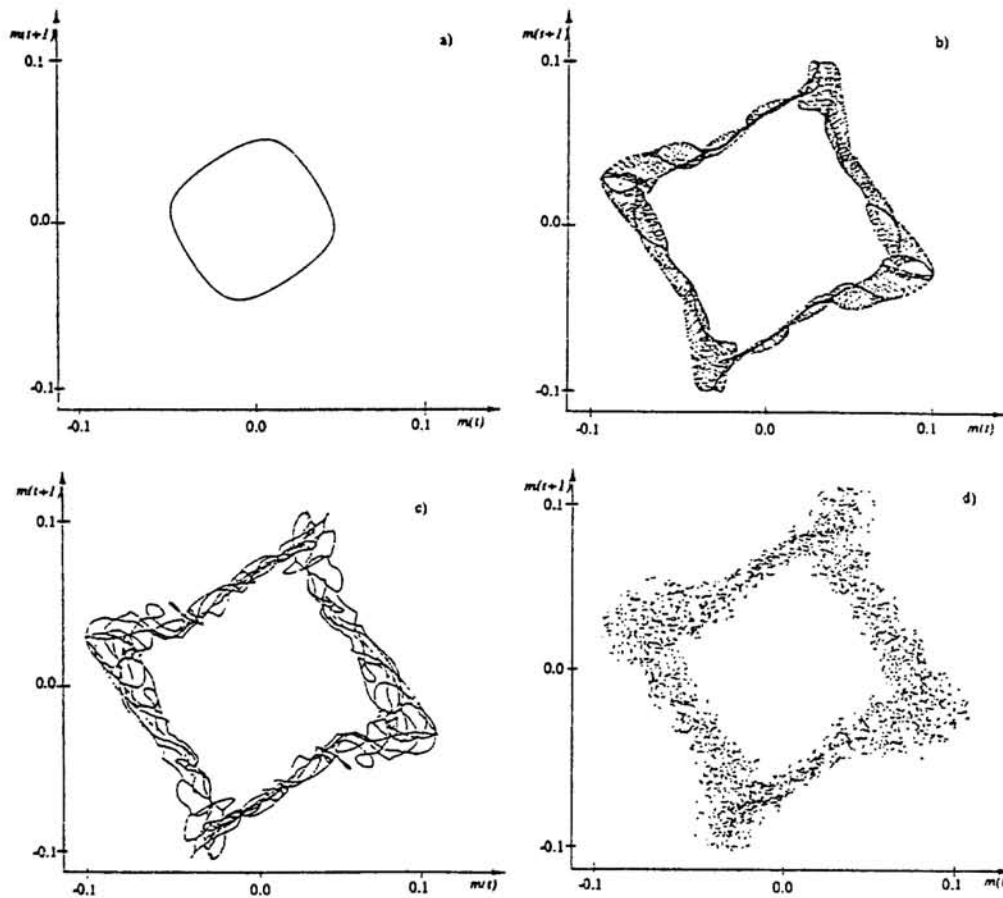

Figure 2: Example of route to chaos when the first bifurcation is a Hopf one.
($N$=128, $K$=16).
 a) After the first bifurcation, the zero fixed point has lost its stability.
  The series of points ($m(t)$, $m(t+1)$) densely covers a cycle ($gJ$=1.0).
 b) After the second Hopf bifurcation: projection of a $T^2$ torus ($gJ$=1.23).
 c) Frequency locking on the $T^2$ torus ($gJ$=1.247).
 d) Chaos ($gJ$=1.26).

When the first bifurcation is a Hopf one (Fig. 2a), it is followed by a second Hopf bifurcation (Fig. 2b). Then there is a frequency locking occuring on the $T^2$ torus born from the second Hopf bifurcation (Fig. 2c), followed by chaos (Fig. 2d). This route is then a quasi-periodic one (Ruelle & Takens, 1971 ; Newhouse et al., 1978). A slightly different feature emerges when the first bifurcation is followed by a stable resonance due to discrete time occuring before the second Hopf bifurcation. Then the limit cycle reduces to periodic points. When the second bifurcation occurs, the resonance persists until chaos is reached.

When the first bifurcation is a pitchfork, it is followed by a Hopf bifurcation for each stable point of the pitchfork (due to the symmetry $x \rightarrow -x$). Then a second Hopf bifurcation occurs followed, via a frequency locking, by chaos. It follows then, despite the pitchfork bifurcation, a quasi-periodicity route. Notice that in this case, we get two symmetric strange attractors. When $gJ$ increases, the two attractors fuse.

For a first bifurcation of flip type, the route followed is like the one described by Bauer & Martienssen (1989). The flip bifurcation leads to an oscillatory system with two states. A first Hopf bifurcation arises followed by a second one leading to a quasi-periodic state, followed by a frequency locking preceeding chaos.

# 5    CONCLUSION

We have presented a type of neural network exhibiting a chaotic behavior when increasing a bifurcation parameter. As in Sompolinsky's model, $gJ$ is *the* control parameter of the network dynamics. The variance of the synaptic weights being normalized, the bifurcation values are nearly independent of the connectivity $K$. The magnitude of dilution is not important for the behavior. The route to chaos by quasi-periodicity seems to be generic. It suggests that such high-dimensional networks behave like low-dimensional dynamical systems. It could be much simpler to control such networks than *a priori* expected.

From a biological point of view, we built our model to provide a tool that could be used to investigate the influence of chaotic dynamics in the cognitive processes in the brain. We clearly chose to simplify the biological complexity in order to understand a complex dynamic. We think that, if chaos plays a role in cognitive processes, it does neither depend on a specific architecture, nor on the exact internal modelling of the biological neuron. However, it could be interesting to introduce some biological caracteristics in the model. The next step will be to study the influence of non-zero entries on the behavior of the system, leading to the modelling of learning in a chaotic network.

**Acknowledgements**

This research has been partly supported by the COGNISCIENCE research program of the C.N.R.S. through PRESCOT, the Toulouse network of searchers in Cognitive Sciences.

## References

M. Bauer & W. Martienssen. (1989) Quasi-Periodicity Route to Chaos in Neural Networks. *Europhys. Lett.* **10**: 427-431.

B. Doyon, B. Cessac, M. Quoy & M. Samuelides. (1993) Control of the Transition to Chaos in Neural Networks with Random Connectivity. *Int. J. Bifurcation and Chaos (in press).*

D. Gallez & A. Babloyantz. (1991) Predictability of human EEG: a dynamical approach. *Biol. Cybern.* **64**: 381-392.

V.I. Girko. (1985) Circular Law. *Theory Prob. Its Appl. (USSR)* **29**: 694-706.

C.M. Gray, P. Koenig, A.K. Engel & W. Singer. (1989) Oscillatory responses in cat visual cortex exhibit intercolumnar synchronisation which reflects global stimulus properties. *Nature* **338**: 334-337.

M. W. Hirsch. (1989) Convergent Activation Dynamics in Continuous Time Networks. *Neural Networks* **2**: 331-349.

S. Newhouse, D. Ruelle & F. Takens. (1978) Occurrence of Strange Axiom *A* Attractors Near Quasi Periodic Flows on $T^m$, $m \geq 3$. *Commun. math. Phys.* **64**: 35-40.

D. Ruelle & F. Takens. (1971) On the nature of turbulence. *Comm. math. Phys.* **20**: 167-192.

C.A. Skarda & W.J. Freeman. (1987) How brains makes chaos in order to make sense of the world. *Behav. Brain Sci.* **10**: 161-195.

H.J. Sommers, A. Crisanti, H. Sompolinsky & Y. Stein. (1988) Spectrum of large random asymmetric matrices. *Phys. Rev. Lett.* **60**: 1895-1898.

H. Sompolinsky, A. Crisanti & H.J. Sommers. (1988) Chaos in random neural networks. *Phys. Rev. Lett.* **61**: 259-262.
